# Multiagent Planning with Factored MDPs

**Carlos Guestrin**
Computer Science Dept
Stanford University
*guestrin@cs.stanford.edu*

**Daphne Koller**
Computer Science Dept
Stanford University
*koller@cs.stanford.edu*

**Ronald Parr**
Computer Science Dept
Duke University
*parr@cs.duke.edu*

## Abstract

We present a principled and efficient planning algorithm for cooperative multiagent dynamic systems. A striking feature of our method is that the coordination and communication between the agents is not imposed, but derived directly from the system dynamics and function approximation architecture. We view the entire multiagent system as a single, large Markov decision process (MDP), which we assume can be represented in a factored way using a dynamic Bayesian network (DBN). The action space of the resulting MDP is the joint action space of the entire set of agents. Our approach is based on the use of factored linear value functions as an approximation to the joint value function. This factorization of the value function allows the agents to coordinate their actions at runtime using a natural message passing scheme. We provide a simple and efficient method for computing such an approximate value function by solving a single linear program, whose size is determined by the interaction between the value function structure and the DBN. We thereby avoid the exponential blowup in the state and action space. We show that our approach compares favorably with approaches based on reward sharing. We also show that our algorithm is an efficient alternative to more complicated algorithms even in the single agent case.

## 1  Introduction

Consider a system where multiple agents, each with its own set of possible actions and its own observations, must coordinate in order to achieve a common goal. We want to find a mechanism for coordinating the agents' actions so as to maximize their joint utility. One obvious approach to this problem is to represent the system as a Markov decision process (MDP), where the "action" is a joint action for all of the agents and the reward is the total reward for all of the agents. Unfortunately, the action space is exponential in the number of agents, rendering this approach impractical in most cases. Alternative approaches to this problem have used local optimization for the different agents, either via reward/value sharing [11, 13] or direct policy search [10].

We present a novel approach based on approximating the joint value function as a linear combination of local value functions, each of which relates only to the parts of the system controlled by a small number of agents. We show how such factored value functions allow the agents to find a globally optimal joint action using a very natural message passing scheme. We provide a very efficient algorithm for computing such a factored approximation to the true value function using a linear programming approach. This approach is of independent interest, as it is significantly faster and compares very favorably to previous approximate algorithms for single agent MDPs. We also compare our multiagent algorithm to the multiagent reward and value sharing algorithms of Schneider et al. [11], showing that our algorithm achieves superior performance which in fact is close to the achievable optimum for this class of problems.

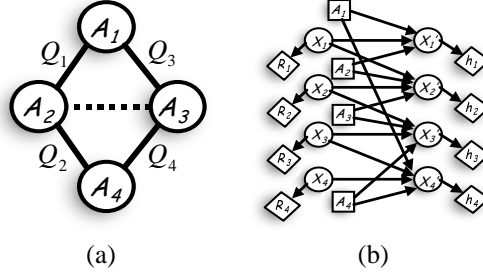

(a)                        (b)

Figure 1: (a) Coordination graph for a 4-agent problem. (b) A DBN for a 4-agent MDP.

## 2 Cooperative Action Selection

We begin by considering a simpler problem of selecting a globally optimal joint action in order to maximize the joint immediate value achieved by a set of agents. Suppose we have a collection of agents, where each agent $j$ chooses an action $A_j$. We use $\mathbf{A}$ to denote $\{A_1, \ldots, A_g\}$. Each agent $j$ has a local Q function $Q_j$, which represents its local contribution to the total utility function. The agents are jointly trying to maximize $Q = \sum_j Q_j$. An agent's local $Q_j$ function might be influenced by its action and those of some other agents; we define the *scope* $\text{Scope}[Q_j] \subset \mathbf{A}$ to be the set of agents whose action influences $Q_j$. Each $Q_j$ may be further decomposed as a linear combination of functions that involve fewer agents; in this case, the complexity of the algorithm may be further reduced.

Our task is to select a joint action $\mathbf{a}$ that maximizes $\sum_j Q_j(\mathbf{a})$. The fact that the $Q_j$ depend on the actions of multiple agents forces the agents to coordinate their action choices. We can represent the coordination requirements of the system using a *coordination graph*, where there is a node for each agent and an edge between two agents if they must directly coordinate their actions to optimize some particular $Q_i$. Fig. 1(a) shows the coordination graph for an example where $Q = Q_1(a_1, a_2) + Q_2(a_2, a_4) + Q_3(a_1, a_3) + Q_4(a_3, a_4)$. A graph structure suggests the use of a *cost network* [5], which can be solved using *non-serial dynamic programming* [1] or a variable elimination algorithm which is virtually identical to variable elimination in a Bayesian network.

The key idea is that, rather than summing all functions and then doing the maximization, we maximize over variables one at a time. Specifically, when maximizing over $a_l$, only summands involving $a_l$ participate in the maximization. Let us begin our optimization with agent 4. To optimize $A_4$, functions $Q_1$ and $Q_3$ are irrelevant. Hence, we obtain:

$$\max_{a_1, a_2, a_3} Q_1(a_1, a_2) + Q_3(a_1, a_3) + \max_{a_4}[Q_2(a_2, a_4) + Q_4(a_3, a_4)].$$

We see that to optimally choose $A_4$, the agent must know the values of $A_2$ and $A_3$. In effect, it is computing a conditional strategy, with a (possibly) different action choice for each action choice of agents 2 and 3. Agent 4 can summarize the value that it brings to the system in the different circumstances using a new function $e_4(A_2, A_3)$ whose value at the point $a_2, a_3$ is the value of the internal max expression. Note that $e_4$ introduces a new induced communication dependency between agents $A_2$ and $A_3$, the dashed line in Fig. 1(a).

Our problem now reduces to computing $\max_{a_1, a_2, a_3} Q_1(a_1, a_2) + Q_3(a_1, a_3) + e_4(a_2, a_3)$, having one fewer agent. Next, agent 3 makes its decision, giving: $\max_{a_1, a_2} Q_1(a_1, a_2) + e_3(a_1, a_2)$, where $e_3(a_1, a_2) = \max_{a_3}[Q_3(a_1, a_3) + e_1(a_2, a_3)]$. Agent 2 now makes its decision, giving $e_2(a_1) = \max_{a_2}[Q_1(a_1, a_2) + e_3(a_1, a_2)]$, and agent 1 can now simply choose the action $a_1$ that maximizes $e_1 = \max_{a_1} e_2(a_1)$.

We can recover the maximizing set of actions by performing the entire process in reverse: The maximizing choice for $e_1$ selects the action $a_1^*$ for agent 1. To fulfill its commitment to agent 1, agent 2 must choose the value $a_2^*$ which maximizes $e_2(a_1^*)$. This, in turn, forces agent 3 and then agent 4 to select their actions appropriately.

In general, the algorithm maintains a set $\mathcal{F}$ of functions, which initially contains $\{Q_1, \ldots, Q_g\}$. The algorithm then repeats the following steps: (1) Select an uneliminated agent $A_l$. (2) Take all $e_1, \ldots, e_L \in \mathcal{F}$ whose scope contains $A_l$. (3) Define a new function $e = \max_{a_l} \sum_j e_j$ and introduce it into $\mathcal{F}$; the scope of $e$ is $\cup_{j=1}^{L} \text{Scope}[e_j] - \{A_l\}$. As above, the maximizing action choices are recovered by sending messages in the reverse direction. The cost of this algorithm is linear in the number of new "function values" introduced, or exponential in the *induced width* of the coordination graph [5]. Furthermore, each agent does not need to communicate directly with every other agent, instead the necessary communication bandwidth will also be the induced width of the graph, further simplifying the coordination mechanism. We note that this algorithm is essentially a special case of the algorithm used to solve influence diagrams with multiple parallel decisions [7] (as is the one in the next section). However, to our knowledge, these ideas have not been applied to the problem of coordinating the decision making process of multiple collaborating agents.

## 3 One-Step Lookahead

We now consider two elaborations to the action selection problem of the previous section. First, we assume that the agents are acting in a space described by a set of discrete state variables, $\mathbf{X} = \{X_1 \ldots X_n\}$, where each $X_i$ takes on values in some finite domain $\text{Dom}(X_i)$. A state $\mathbf{x}$ defines a value $x_i \in \text{Dom}(X_i)$ for each variable $X_i$. The scope of the local $Q_j$ functions that comprise the value can include both action choices and state variables. We assume that the agents have full observability of the relevant state variables, so by itself, this extension is fairly trivial: The $Q_j$ functions define a conditional cost network. Given a particular state $x_1, \ldots, x_n$, the agents instantiate the state variables and then solve the cost network as in the previous section. However, we note that the agents do not actually need to have access to all of the state variables: agent $j$ only needs to observe the variables that are in the scope of its local $Q_j$ function, thereby decreasing considerably the amount of information each agent needs to observe.

The second extension is somewhat more complicated: We assume that the agents are trying to maximize the sum of an immediate reward and a value that they expect to receive one step in the future. We describe the dynamics of the system $G$ using a *dynamic decision network (DDN)* [4]. Let $X_i$ denote the variable $X_i$ at the current time and $X_i'$ the variable at the next step. The *transition graph* of a DDN is a two-layer directed acyclic graph $G$ whose nodes are $\{A_1, \ldots, A_g, X_1, \ldots, X_n, X_1', \ldots, X_n'\}$, and where only nodes in $\mathbf{X}'$ have parents. We denote the parents of $X_i'$ in the graph by *Parents*$(X_i')$. For simplicity of exposition, we assume that *Parents*$(X_i') \subseteq \mathbf{X} \cup \mathbf{A}$. (This assumption can be relaxed, but our algorithm becomes somewhat more complex.) Each node $X_i'$ is associated with a *conditional probability distribution (CPD)* $P(X_i' \mid \textit{Parents}(X_i'))$. The transition probability $P(\mathbf{x}' \mid \mathbf{x}, \mathbf{a})$ is then defined to be $\prod_i P(x_i' \mid \mathbf{u}_i)$, where $\mathbf{u}_i$ is the value in $\mathbf{x}, \mathbf{a}$ of the variables in *Parents*$(X_i')$. The immediate rewards are a set of functions $r_1, \ldots, r_g$, and the next-step values are a set of functions $h_1, \ldots, h_g$. Here, we assume that both $r_i$'s and $h_i$'s are functions that depend only on a small set of variables.

Fig. 1(b) shows a DDN for a simple four-agent problem, where the ovals represent the variables $X_i$ and the rectangles the agent actions. The diamond nodes in the first time step represent the immediate reward, while the $h$ nodes in the second time step represent the future value associated with a subset of the state variables.

For any setting of the state variables, $\mathbf{x}$, the agents aim to maximize $\mathcal{V}(\mathbf{x}) = \max_{a_1, \ldots, a_g} \sum_{j=1}^{g} [r_j(\mathbf{x}, \mathbf{a}) + \sum_{\mathbf{x}'} P(\mathbf{x}' \mid \mathbf{x}, \mathbf{a}) h_j(\mathbf{x}')]$, i.e., the immediate reward plus the expected value of the next state. The expectation is a summation over an exponential number of future states. As shown in [8], this can be simplified substantially. For example, if we consider the function $h_1(X_1')$ in Fig. 1(b), we can see that its expected value is a function only of $X_1, A_1, A_2$. More generally, we define $g_j(\mathbf{x}, \mathbf{a}) = \sum_{\mathbf{x}'} P(\mathbf{x}' \mid \mathbf{x}, \mathbf{a}) h_j(\mathbf{x}')$. Recall our assumption that the scope of each $h_j$ is only a small subset of variables $\mathbf{C}_j$. Then,

the scope of $g_j$ is $\Gamma(\mathbf{C}_j) = \cup_{X_i' \in \mathbf{C}_j'} Parents(X_i')$. Specifically, $g_j(\mathbf{x}, \mathbf{a}) = \sum_{\mathbf{c}_j'} P(\mathbf{c}_j' \mid \mathbf{z_j}) \mathbf{h_j}(\mathbf{c_j'})$, where $\mathbf{z}_j$ is a value of $\Gamma(\mathbf{C}_j)$. Note that the cost of the computation depends linearly on $|\mathrm{Dom}(\Gamma(\mathbf{C}_j))|$, which depends on $\mathbf{C}_j$ (the scope of $h_j$) and on the complexity of the process dynamics.

By replacing the expectation with the backprojection, we can once again generate a set of local $Q$ functions $Q_j = r_j + g_j$, and apply our coordination graph algorithm unchanged.

## 4  Markov Decision Processes

We now turn our attention to the substantially more complex case where the agents are acting in a dynamic environment, and are jointly trying to maximize their expected long-term return. The *Markov Decision Process (MDP)* framework formalizes this problem. An MDP is defined as a 4-tuple $(\mathbf{X}, \mathcal{A}, R, P)$ where: $\mathbf{X}$ is a finite set of $N = |\mathbf{X}|$ states; $\mathcal{A}$ is a set of actions; $R$ is a *reward function* $R : \mathbf{X} \times \mathcal{A} \mapsto \mathbb{R}$, such that $R(\mathbf{x}, \mathbf{a})$ represents the reward obtained in state $\mathbf{x}$ after taking action $\mathbf{a}$; and $P$ is a *Markovian transition model* where $P(\mathbf{x}' \mid \mathbf{x}, \mathbf{a})$ represents the probability of going from state $\mathbf{x}$ to state $\mathbf{x}'$ with action $\mathbf{a}$. We assume that the MDP has an infinite horizon and that future rewards are discounted exponentially with a discount factor $\gamma \in [0, 1)$.

The optimal value function $\mathcal{V}^*$ is defined so that the value of a state must be the maximal value achievable by any action at that state. More precisely, we define $Q_{\mathcal{V}}(\mathbf{x}, \mathbf{a}) = R(\mathbf{x}, \mathbf{a}) + \gamma \sum_{\mathbf{x}'} P(\mathbf{x}' \mid \mathbf{x}, \mathbf{a}) \mathcal{V}(\mathbf{x}')$, and the *Bellman operator* $\mathcal{T}^*$ to be $\mathcal{T}^* \mathcal{V}(\mathbf{x}) = \max_{\mathbf{a}} Q_{\mathcal{V}}(\mathbf{x}, \mathbf{a})$. The optimal value function $\mathcal{V}^*$ is the fixed point of $\mathcal{T}^*$: $\mathcal{V}^* = \mathcal{T}^* \mathcal{V}^*$. A stationary policy $\pi$ for an MDP is a mapping $\pi : \mathbf{X} \mapsto A$, where $\pi(\mathbf{x})$ is the action the agent takes at state $\mathbf{x}$. For any value function $\mathcal{V}$, we can define the policy obtained by acting greedily relative to $\mathcal{V}$: $Greedy(\mathcal{V})(\mathbf{x}) = \arg\max_{\mathbf{a}} Q_{\mathcal{V}}(\mathbf{x}, \mathbf{a})$. The greedy policy relative to the optimal value function $\mathcal{V}^*$ is the optimal policy $\pi^* = Greedy(\mathcal{V}^*)$.

There are several algorithms for computing the optimal policy. One is via linear programming. Numbering the states in $\mathbf{X}$ as $\mathbf{x}_1, \dots, \mathbf{x}_N$, our variables are $V_1, \dots, V_N$, where $V_i$ represents $\mathcal{V}(\mathbf{x}_i)$. The LP is:

Minimize: $\quad \sum_i \alpha(\mathbf{x}_i) V_i$ ;
Subject to: $\quad V_i \geq R(\mathbf{x}_i, \mathbf{a}) + \gamma \sum_l P(\mathbf{x}_l' \mid \mathbf{x}_i, \mathbf{a}) V_k \quad \forall \mathbf{x}_i \in \mathbf{X}, \mathbf{a} \in \mathcal{A}.$

The state relevance weights $\alpha$ are *any* convex weights, with $\alpha(\mathbf{x}) > 0$ and $\sum_{\mathbf{x}} \alpha(\mathbf{x}) = 1$.

In our setting, the state space is exponentially large, with one state for each assignment $\mathbf{x}$ to $\{X_1, \dots, X_n\}$. We use the common approach of restricting attention to value functions that are compactly represented as a linear combination of *basis functions* $H = \{h_1, \dots, h_k\}$. A *linear value function* over $H$ is a function $\mathcal{V}$ that can be written as $\mathcal{V}(\mathbf{x}) = \sum_{j=1}^k w_j h_j(\mathbf{x})$ for some coefficients $\mathbf{w} = (w_1, \dots, w_k)'$.

The LP approach can be adapted to use this value function representation [12]:

Variables: $\quad w_1, \dots, w_k$ ;
Minimize: $\quad \sum_j \alpha_j w_j$ ;
Subject to: $\quad \sum_{j=1}^k w_j h_j(\mathbf{x}_i) \geq R(\mathbf{x}_i, \mathbf{a}) +$
$\qquad\qquad \gamma \sum_{\mathbf{x}_l'} P(\mathbf{x}_l' \mid \mathbf{x}_i, \mathbf{a}) \sum_{j=1}^k w_j h_j(\mathbf{x}_l'); \quad \forall \mathbf{x}_i \in \mathbf{X}, \mathbf{a} \in \mathcal{A}.$

Where $\alpha_j = \sum_{\mathbf{x}_i} \alpha(\mathbf{x}_i) h_j(\mathbf{x}_i)$. This transformation has the effect of reducing the number of free variables in the LP to $k$ but the number of constraints remains $|\mathbf{X}| \times |\mathcal{A}|$. There is, in general, no guarantee as to the quality of the approximation $\sum_{j=1}^k w_j h_j$, but recent work of de Farias and Van Roy [3] provides some analysis of the error relative to that of the best possible approximation in the subspace, and some guidance as to selecting the $\alpha$'s so as to improve the quality of the approximation.

## 5 Factored MDPs

*Factored MDPs* [2] allow the representation of large structured MDPs by using a dynamic Bayesian network to represent the transition model. Our representation of the one-step transition dynamics in Section 3 is precisely a factored MDP, where we factor not only the states but also the actions. In [8], we proposed the use of *factored linear value functions* to approximate the value function in a factored MDP. These value functions are a weighted linear combination of basis functions, as above, but each basis function is restricted to depend only on a small subset of state variables. The $h$ functions in Fig. 1(b) are an example. If we had a value function $\mathcal{V}$ represented in this way, then we could use our algorithm of Section 3 to implement *Greedy*($\mathcal{V}$) by having the agents use our message passing coordination algorithm at each step. (Here we have only one function $h$ per agent, but our approach extends trivially to the case of multiple $h$ functions.)

In previous work [9, 6], we presented algorithms for computing approximate value functions of this form for factored MDPs. These algorithms can circumvent the exponential blowup in the number of state variables, but explicitly enumerate the action space of the MDP, making them unsuitable for the exponentially large action space in multiagent MDPs. We now provide a novel algorithm based on the LP of the previous section. In particular, we show how we can solve this LP exactly in closed form, without explicitly enumerating the exponentially many constraints.

Our first task is to compute the coefficients $\alpha_j$ in the objective function. Note that, $\alpha_j = \sum_{\mathbf{x}} \alpha(\mathbf{x}) h_j(\mathbf{x}) = \sum_{\mathbf{c_j}} \alpha(\mathbf{c_j}) \mathbf{h_j}(\mathbf{c_j})$, as basis $h_j$ has scope restricted to $\mathbf{C}_j$. Here, $\alpha(\mathbf{c_j})$ represents the marginal of the state relevance weights $\alpha$ over $\mathbf{C}_j$. Thus, the coefficients $\alpha_j$ can be pre-computed efficiently if $\alpha$ is represented compactly by its marginals $\alpha(\mathbf{C}_j)$. Our experiments used uniform weights $\alpha(\mathbf{x}) = \frac{1}{|\mathbf{X}|}$, thus, $\alpha(\mathbf{c_j}) = \frac{1}{|\mathbf{C_j}|}$.

We must now deal with the exponentially large constraint set. Using the backprojection from Section 3, we can rewrite our constraints as:

$$\sum_{j=1}^{k} w_j h_j(\mathbf{x}) \geq R(\mathbf{x}, \mathbf{a}) + \gamma \sum_{j=1}^{k} w_j g_j(\mathbf{x}, \mathbf{a}); \ \forall \mathbf{x} \in \mathbf{X}, \mathbf{a} \in \mathcal{A}$$

where $g_j(\mathbf{x}, \mathbf{a}) = \sum_{\mathbf{x}'} P(\mathbf{x}' \mid \mathbf{x}, \mathbf{a}) h_j(\mathbf{x}')$. Note that this exponentially large set of linear constraints can be replaced by a single, equivalent, non-linear constraint:

$$0 \geq \max_{\mathbf{x}, \mathbf{a}} R(\mathbf{x}, \mathbf{a}) + \sum_{j=1}^{k} w_j [\gamma g_j(\mathbf{x}, \mathbf{a}) - h_j(\mathbf{x})].$$

In a factored MDP, the reward function $R$ is represented as the sum of local rewards $\sum_i r_i$. Furthermore, the basis $h_j$ and its backprojection $g_j$ are also functions that depend only on a small set of variables. Thus, the right side of the constraint can be viewed as the sum of restricted scope functions parameterized by $\mathbf{w}$. For a fixed $\mathbf{w}$, we can compute the maximum over $\{\mathbf{x}, \mathbf{a}\}$ using a cost network, as in Section 2. If $\mathbf{w}$ is not specified, the maximization induces a family of cost networks parameterized by $\mathbf{w}$. As we showed in [6], we can turn this cost network into a compact set of LP constraints on the free variable $\mathbf{w}$.

More generally, suppose we wish to enforce the constraint $0 \geq \max_{\mathbf{y}} F^{\mathbf{w}}(\mathbf{y})$, where $F^{\mathbf{w}}(\mathbf{y}) = \sum_j f_j^{\mathbf{w}}(\mathbf{y})$ such that each $f_j$ has a restricted scope. Here, the superscript $\mathbf{w}$ indicates that each $f_j$ might be multiplied by a weight $w$, but this dependency is linear. Consider the cost network used to maximize $F^{\mathbf{w}}$; let $e$ by any function used in the network, including the original $f_j$'s, and let $\mathbf{Z}$ be its scope. For any assignment $\mathbf{z}$ to $\mathbf{Z}$, we introduce a variable $u_{\mathbf{z}}^e$, whose value represents $e(\mathbf{z})$, into the linear program. For the initial functions $f_j^{\mathbf{w}}$, we include the constraint that $u_{\mathbf{z}}^{f_j} = f_j^{\mathbf{w}}(\mathbf{z})$. As $f_j^{\mathbf{w}}$ is linear in $\mathbf{w}$, this constraint is linear in the LP variables. Now, consider a new function $e$ introduced into $\mathcal{F}$ by eliminating a variable $Y_l$. Let $e_1, \ldots, e_L$ be the functions extracted from $\mathcal{F}$, with scope $\mathbf{Z}_1, \ldots, \mathbf{Z}_L$

- **Offline:**
  1. Select a set of restricted scope basis functions $\{h_1, \ldots, h_k\}$.
  2. Apply efficient LP-based approximation algorithm offline (Section 5) to compute co-efficients $\{w_1, \ldots, w_k\}$ of the approximate value function $\mathcal{V} = \sum_j w_j h_j$.
  3. Use the one-step lookahead planning algorithm (Section 3) with $\mathcal{V}$ as a value function estimate to compute local $Q_j$ functions for each agent.

- **Online:** At state $\mathbf{x}$:
  1. Each agent $j$ instantiates $Q_j$ with values of state variables in scope of $Q_j$.
  2. Agents apply coordination graph algorithm (Section 2) with local $Q_j$ functions to coordinate approximately optimal global action.

Figure 2: Algorithm for multiagent planning with factored MDPs

respectively. As in the cost network, we want that $u_{\mathbf{z}}^e = \max_{y_l} [\sum_{j=1}^{L} u_{\mathbf{z}_j}^{e_j}]$ where $\mathbf{z}_j$ is the value of $\mathbf{Z}_j$ in the instantiation $(\mathbf{z}, y_l)$. We enforce this by introducing a set of constraints into our LP: $u_{\mathbf{z}}^e \geq \sum_{j=1}^{L} u_{\mathbf{z}_j}^{e_j} \quad \forall y_l$. The last function generated in the elimination, $e_m$, has an empty domain. We introduce the additional constraint $0 \geq u^{e_m}$, which is equivalent to the global constraint $0 \geq \max_{\mathbf{y}} F^{\mathbf{w}}(\mathbf{y})$.

In the case of cooperative multiagent MDPs, the actions of the individual agents become variables in the cost network, so that the set $\mathbf{Y}$ is simply $\mathbf{X} \cup \mathbf{A}$. The functions $f_j^{\mathbf{w}}$ are simply the local functions corresponding to the rewards $r_j$, the bases $h_j$ and their backprojections $g_j$. We can thus write down constraints that enforce $\sum_{j=1}^{k} w_j h_j(\mathbf{x}) \geq R(\mathbf{x}, \mathbf{a}) + \gamma \sum_{\mathbf{x}'} P(\mathbf{x}' \mid \mathbf{x}, \mathbf{a}) \sum_{j=1}^{k} w_j h_j(\mathbf{x}')$ over the entire exponential state space and joint action space using a number of constraints which is only exponential in the induced tree width of the cost network, rather than exponential in the number of actions and state variables in the problem.

A traditional single agent is, of course, a special case of the multiagent case. The LP approach described in this section provides an attractive alternative to the methods described in [9] and [6]. In particular, our approach requires that we solve a single LP, whose size is essentially the size of the cost network. The approach of [6] (which is substantially more efficient than that of [9]) requires that we solve an LP for each step in policy iteration, and each LP contains constraints corresponding to multiple cost networks (whose number depends on the complexity of the policy representation). Furthermore, the LP approach eliminates the restrictions on the action model made in [9, 6].

Our overall algorithm for multiagent planning with factored MDPs in shown in Fig. 2.

## 6   Experimental results

We first validate our approximate LP approach by comparing the quality of the solution to the approximate policy iteration (PI) approach of [6]. As the approximate PI algorithm is not able to deal with the exponentially large action spaces of multiagent problems, we compare these two approaches on the single agent SysAdmin problem presented in [6], on a unidirectional ring network of up to 32 machines (over 4 billion states). As shown in Fig. 3(b), our new approximate LP algorithm for factored MDPs is significantly faster than the approximate PI algorithm. In fact, approximate PI with single-variable basis functions variables is more costly than the LP approach using basis functions over consecutive triples of variables. As shown in Fig. 3(c), for singleton basis functions, the approximate PI policy obtains slightly better performance for some problem sizes. However, as we increase the number of basis functions for the approximate LP formulation, the value of the resulting policy is much better. Thus, in this problem, our new approximate linear programming formulation allows us to use more basis functions and to obtain a resulting policy of higher value, while still maintaining a faster running time.

We constructed a multiagent version of the SysAdmin problem, applied to various net-

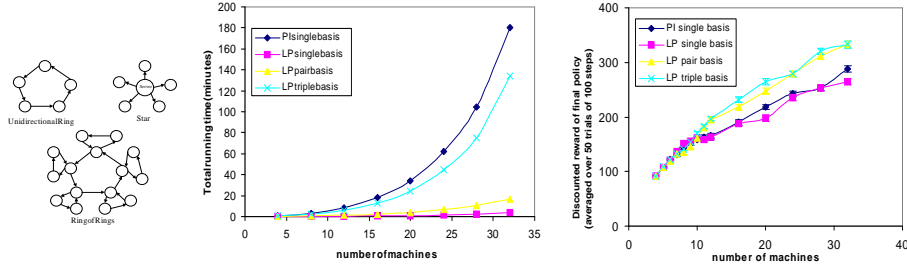

Figure 3: (a) Network topologies used in our experiments. Graphs: Approximate LP versus approximate PI on single agent SysAdmin on unidirectional ring: (b) running time; (c) estimated value of policy.

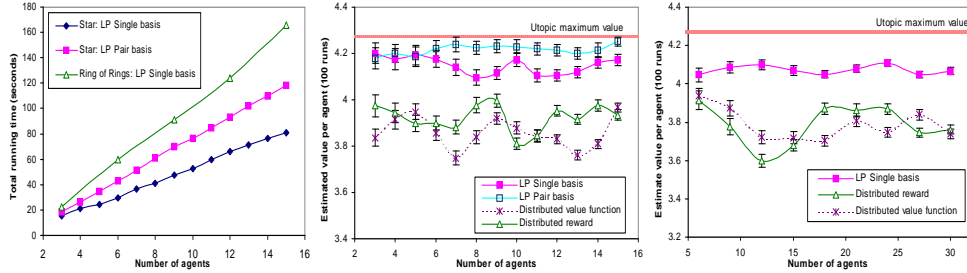

Figure 4: (a) Running time for approximate LP for increasing number of agents. Policy performance of approximate LP and DR/DRF: (b) on "star"; (c) on "ring of rings".

work architectures shown in Fig. 3(a). Each machine is associated with an agent $A_i$ and two variables: Status $S_i \in \{good, faulty, dead\}$, and Load $L_i \in \{idle, loaded, process successful\}$. A dead machine increases the probability that its neighbors will become faulty and die. The system receives a reward of 1 if a process terminates successfully. If the Status is faulty, processes take longer to terminate. If the machine dies, the process is lost. Each agent $A_i$ must decide whether machine $i$ should be rebooted, in which case the Status becomes good and any running process is lost. For a network of $n$ machines, the number of states in the MDP is $9^n$ and the joint action space contains $2^n$ possible actions, e.g., a problem with 30 agents has over $10^{28}$ states and a billion possible actions.

We implemented the factored approximate linear programming and the message passing coordination algorithms in Matlab, using CPLEX as the LP solver. We experimented with two types of basis functions: "single", which contains an indicator basis function for each value of each $S_i$ and $L_i$; and "pair" which, in addition, contains indicators over joint assignments of the Status variables of neighboring agents. We use $\gamma = 0.95$.

As shown in Fig. 4(a), the total running time of the algorithm grows linearly in the number of agents, for each fixed network and basis type. This is the expected asymptotic behavior, as each problem has a fixed induced tree width of the cost network. (The induced tree width for pair basis on the "ring of rings" problem was too large.)

For comparison, we also implemented the distributed reward (DR) and distributed value function (DRF) algorithms of Schneider et al. [11]. Here we used 10000 learning iterations, with learning and exploration rates starting at 0.1 and 1.0 respectively and a decaying schedule after 5000 iterations; the observations for each agent were the status and load of its machine. The results of the comparison are shown in Fig. 4(b) and (c). We also computed a utopic upper bound on the value of the optimal policy by removing the (negative) effect of the neighbors on the status of the machines. This is a loose upper bound, as a dead neighbor increases the probability of a machine dying by about 50%. For both network topologies tested, the estimated value of the approximate LP solution using single basis was significantly higher than that of the DR and DRF algorithms. Note that the single

basis solution requires no coordination when acting, so this is a "fair" comparison to DR and DRF which also do not communicate while acting. If we allow for pair bases, which implies agent communication, we achieve a further improvement in terms of estimated value. The policies obtained tended to be intuitive: e.g., for the "star" topology with pair basis, if the server becomes faulty, it is rebooted even if loaded. but for the clients, the agent waits until the process terminates or the machine dies before rebooting.

## 7 Conclusion

We have provided principled and efficient approach to planning in multiagent domains. Rather than placing *a priori* restrictions on the communication structure between agents, we first choose the form of an approximate value function and derive the optimal communication structure given the value function architecture. This approach provides a unified view of value function approximation and agent communication. We use a novel linear programming technique to find an approximately optimal value function. The inter-agent communication and the LP avoid the exponential blowup in the state and action spaces, having computational complexity dependent, instead, upon the induced tree width of the coordination graph used by the agents to negotiate their action selection. By exploiting structure in both the state and action spaces, we can deal with considerably larger MDPs than those described in previous work. In a family of multiagent network administration problems with over $10^{28}$ states and over a billion actions, we have demonstrated near optimal performance which is superior to *a priori* reward or value sharing schemes. We believe the methods described herein significantly further extend the efficiency, applicability and general usability of factored value functions and models for the control of dynamic systems.

**Acknowledgments:** This work was supported by ONR under MURI "Decision Making Under Uncertainty", the Sloan Foundation, and the first author was also supported by a Siebel scholarship.

## References

[1] U. Bertele and F. Brioschi. *Nonserial Dynamic Programming*. Academic Press, 1972.

[2] C. Boutilier, T. Dean, and S. Hanks. Decision theoretic planning: Structural assumptions and computational leverage. *Journal of Artificial Intelligence Research*, 11:1 – 94, 1999.

[3] D.P. de Farias and B. Van Roy. The linear programming approach to approximate dynamic programming. *submitted to the IEEE Transactions on Automatic Control*, January 2001.

[4] T. Dean and K. Kanazawa. A model for reasoning about persistence and causation. *Computational Intelligence*, 5(3):142–150, 1989.

[5] R. Dechter. Bucket elimination: A unifying framework for reasoning. *Artificial Intelligence*, 113(1–2):41–85, 1999.

[6] C. Guestrin, D. Koller, and R. Parr. Max-norm projections for factored MDPs. In *Proc. 17th IJCAI*, 2001.

[7] F. Jensen, F. Jensen, and S. Dittmer. From influence diagrams to junction trees. In *Uncertainty in Artificial Intelligence: Proceedings of the Tenth Conference*, pages 367–373, Seattle, Washington, July 1994. Morgan Kaufmann.

[8] D. Koller and R. Parr. Computing factored value functions for policies in structured MDPs. In *Proceedings of the Sixteenth International Joint Conference on Artificial Intelligence (IJCAI-99)*. Morgan Kaufmann, 1999.

[9] D. Koller and R. Parr. Policy iteration for factored MDPs. In *Proc. 16th UAI*, 2000.

[10] L. Peshkin, N. Meuleau, K. Kim, and L. Kaelbling. Learning to cooperate via policy search. In *Proc. 16th UAI*, 2000.

[11] J. Schneider, W. Wong, A. Moore, and M. Riedmiller. Distributed value functions. In *Proc. 16th ICML*, 1999.

[12] P. Schweitzer and A. Seidmann. Generalized polynomial approximations in Markovian decision processes. *Journal of Mathematical Analysis and Applications*, 110:568 – 582, 1985.

[13] D. Wolpert, K. Wheller, and K. Tumer. General principles of learning-based multi-agent systems. In *Proc. 3rd Agents Conference*, 1999.